# A Summating, Exponentially-Decaying CMOS Synapse for Spiking Neural Systems

**Rock Z. Shi**[1,2] **and Timothy Horiuchi**[1,2,3]
[1]Electrical and Computer Engineering Department
[2]Institute for Systems Research
[3]Neuroscience and Cognitive Science Program
University of Maryland, College Park, MD 20742
`rshi@glue.umd.edu,timmer@isr.umd.edu`

## Abstract

Synapses are a critical element of biologically-realistic, spike-based neural computation, serving the role of communication, computation, and modification. Many different circuit implementations of synapse function exist with different computational goals in mind. In this paper we describe a new CMOS synapse design that separately controls quiescent leak current, synaptic gain, and time-constant of decay. This circuit implements part of a commonly-used kinetic model of synaptic conductance. We show a theoretical analysis and experimental data for prototypes fabricated in a commercially-available $1.5\mu m$ CMOS process.

## 1 Introduction

Synapses are a critical element in spike-based neural computation. There are perhaps as many different synapse circuit designs in use as there are brain areas being modeled. This diversity of circuits reflects the diversity of the synapse's computational function. In many computations, a narrow, square pulse of current is all that is necessary to model the synaptic current. In other situations, a longer post-synaptic current profile is desirable to extend the effects of extremely short spike durations (e.g., in address-event systems [1],[2], [3], [4]), or to create a specific time window of interaction (e.g., for coincidence detection or for creating delays [5]).

Temporal summation or more complex forms of inter-spike interaction are also important areas of synaptic design that focus on the response to high-frequency stimulation. Recent designs for fast-synaptic depression [6], [7], [8] and time-dependent plasticity [9], [10] are good examples of this where some type of memory is used to create interaction between incoming spikes. Even simple summation of input current can be very important in address-event systems where a common strategy to reduce hardware is to have a single synapse circuit mimic inputs from many different cells. A very popular design for this purpose is the "current-mirror synapse" [4] that is used extensively in its original form or in new extended forms [6], [8] to expand the time course of current and to provide summation for high-frequency spiking. This circuit is simple, compact, and stable, but couples the leak, part of the synaptic gain, and the decay "time-constant" in one control parameter. This is restrictive and often more control is desirable. Alternatively, the same components can be

arranged to give the user manual-control of the decay to produce a true exponential decay when operating in the subthreshold region (see Figure 7 (b) of [11]). This circuit, however, does not provide good summation of multiple synaptic events.

In this paper we describe a new CMOS synapse circuit, that utilizes current-mode feedback to produce a first-order dynamical system. In the following sections, we describe the kinetic model of synaptic conductance, describe the circuit implementation and function, provide a theoretical analysis and finally compare our theory against testing results. We also discuss the use of this circuit in various neuromorphic system contexts and conclude with a discussion of the circuit synthesis approach.

## 2 Proposed synapse model

We consider a network of spiking neurons, each of which is modeled by the integrate-and-fire model or the slightly more generous Spike Response Model (e.g. [12]). Synaptic function in such neural networks are often modeled as a time-varying current. The functional form of this current could be a $\delta$ function, or a limited jump at the time of the spike followed by an exponential decay. Perhaps the most widely used function in detailed computational models is the $\alpha$-function, a function of the form $\frac{t}{\tau}e^{-\frac{t}{\tau}}$, introduced by [13].

A more general and practical framework is the neurotransmitter kinetics description proposed by Destexhe et al. [14]. This approach can synthesize a complete description of synaptic transmission, as well as give an analytic expression for a post-synaptic current in some simplified schemes. For a two-state ligand-gated channel model, the neurotransmitter molecules, T, are taken to bind to post-synaptic receptors modeled by the first order kinetic scheme [15]:

$$R + T \underset{\beta}{\overset{\alpha}{\rightleftharpoons}} TR^*$$ (1)

where R and $TR^*$ are the unbound and the bound form of the post-synaptic receptor, respectively. $\alpha$ and $\beta$ are the forward and backward rate constants for transmitter binding. In this model, the fraction of bound receptors, r, is described by the equation:

$$\frac{dr}{dt} = \alpha[T](1 - r) - \beta r$$ (2)

If the transmitter concentration [T] can be modeled as a short pulse, then r(t) in (2) is a first order linear differential equation.

We propose a synapse model that can be implemented by a CMOS circuit working in the subthreshold region. Our model matches Destexhe et al.'s equations for the time-dependent conductance, although we assume a fixed driving potential. In our synapse model, the action potential is modeled as a narrow digital pulse. The pulse width is assumed to be a fixed value $t_{pw}$, however, in practice $t_{pw}$ may vary slightly from pulse to pulse.

Figure 1 illustrates the synaptic current response to a single pulse in such a model:

1. A presynaptic spike occurs at $t_j$, during the pulse, the post-synaptic current is modeled by:

$$i_{syn}(t) = i_{syn}(\infty) + (i_{syn}(t_j) - i_{syn}(\infty))e^{-\frac{t-t_j}{\tau_r}}$$ (3)

2. After the presynaptic pulse terminated at time $t_j + t_{pw}$, the post-synaptic current is modeled by:

$$i_{syn}(t) = i_{syn}(t_j + t_{pw})e^{-\frac{t-t_j-t_{pw}}{\tau_d}}$$ (4)

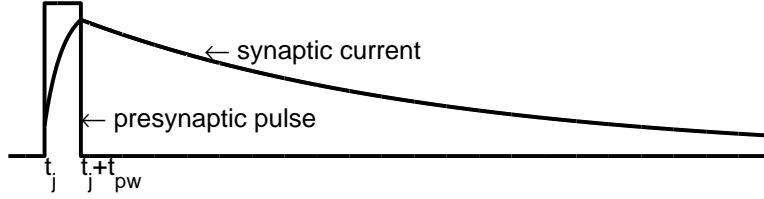

Figure 1: Synapse model. The action potential (spike) is modeled as a pulse with width $t_{pw}$. The synapse is modeled as first order linear system with synaptic current response described by Equations (3) and (4)

# 3   CMOS circuit synthesis and analysis

## 3.1   The synthesis approach

Lazzaro [11] presents a very simple, compact synapse circuit that has an exponentially-decaying synaptic current after each spike event. The synaptic current always resets to the maximum current value during the spike and is not suitable for the summation of rapid bursts of spikes. Another simple and widely used synapse is the current-mirror synapse that has its own set of practical problems related to the coupling of gain, time constant, and offset parameters. Our circuit is synthesized from the clean exponential decay from Lazzaro's synapse and concepts from log domain filtering [16], [17] to convert the nonlinear characteristic of the current mirror synapse into an externally-linear, time-invariant system [18].

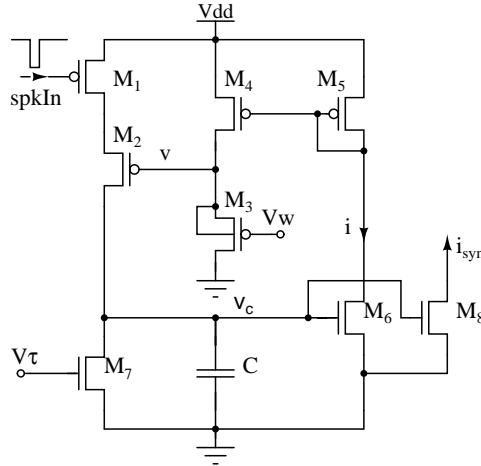

Figure 2: The proposed synapse circuit. The pin "spkIn" receives the spike input with negative logic. The pin "$i_{syn}$" is the synaptic current output. There are two control parameters. The input voltage Vw adjusts the weight of the synapse and the input voltage $V_\tau$ sets the time constant. The transistors sizes are: $S_1 = 2.4\mu m/1.6\mu m$, $S_2 = 8\mu m/4\mu m$, $S_3 = 10\mu m/4\mu m \times 4$, $S_4 = 4\mu m/4\mu m$, $S_5 = 4\mu m/4\mu m$, $S_6 = 4\mu m/4\mu m$, $S_7 = 4\mu m/4\mu m$, $S_8 = 10\mu m/4\mu m \times 20$. The bodies of NMOS transistors are connected to ground, and the bodies of PMOS transistors are connected to Vdd except for $M_3$.

## 3.2 Basic circuit description

The synapse circuit consists of eight transistors and one capacitor as shown in Figure 2. All transistors are operated in the subthreshold region. Input voltage spikes are applied through an inverter (not shown), onto the gate of the PMOS $M_1$. $V_\tau$ sets the current through $M_7$ that determines the time constant of the output synaptic current as will be shown later. Vw controls the magnitude of the synaptic current, so it determines the synaptic weight. The voltage on the capacitor is converted to a current by transistor $M_6$, sent through the current mirror $M_4 - M_5$, and into the source follower $M_3 - M_4$. The drain current of $M_8$, a scaled copy of current through $M_6$ produces an inhibitory current. A simple PMOS transistor with the same gate voltage as $M_5$ can provide an excitatory synaptic current.

## 3.3 Circuit analysis

We perform an analysis of the circuit by studying its response to a single spike. Assuming a long transistor so that the Early effect can be neglected, the behavior of a NMOS transistor working in the subthreshold region can be described by [19], [20]

$$i_{ds} = SI_{0n}e^{\frac{\kappa_n v_{gs}}{V_T}} e^{\frac{(1-\kappa_n)v_{bs}}{V_T}} \left(1 - e^{\frac{-v_{ds}}{V_T}}\right) \tag{5}$$

where $V_T = KT/q$ is the thermal voltage, $I_{0n}$ is a positive constant current when $V_{gs} = V_{bs} = 0$, and $S = \frac{W}{L}$ is the ratio of the transistor width and length. $0 < \kappa_n < 1$ is a parameter specific to the technology, and we will assume it is constant in this analysis. We assume that all transistors are operating in saturation ($v_{ds} > 4V_T$). We also neglect any parasitic capacitances.

The PMOS source follower $M_3 - M_4$ is used as a level shifter. Detailed discussion on use of source followers in the subthreshold region has been discussed in [21]. Combined with a current mirror $M_4 - M_5$, this sub-circuit implements a logarithmic relationship between i and v (as labeled in Figure 2):

$$v = V_w + \frac{V_T}{\kappa_p} \ln\left(\frac{i}{I_{0p}} \frac{S_4}{S_3 S_5}\right) \tag{6}$$

Consistent with the translinear principle, this logarithmic relationship will make the current through $M_2$ proportional to $\frac{1}{i}$.

For simplicity, we assume a spike begins at time t=0, and the initial voltage on the capacitor C is $v_c(0)$. The spike ends at time $t = t_{pw}$. When the spike input is on ($0 < t < t_{pw}$), the dynamics of the circuit for a step input is governed by

$$C\frac{dv_c(t)}{dt} = \frac{S_2 S_3 S_5 I_{op}^2}{S_4 S_6 I_{0n}} e^{\frac{\kappa_p (V_{dd} - V_w)}{V_T}} e^{\frac{-\kappa_n v_c(t)}{V_T}} - I_\tau \tag{7}$$

$$I_\tau = S_7 I_{on} e^{\frac{\kappa_n V_\tau}{V_T}} \tag{8}$$

With the aid of transformation

$$i_{syn}(t) = S_8 I_{on} e^{\frac{\kappa_n v_c(t)}{V_T}} \tag{9}$$

Equation (7) can be changed into a linear ordinary differential equation for $i_{syn}(t)$:

$$\frac{di_{syn}(t)}{dt} + \frac{\kappa_n I_\tau}{CV_T} i_{syn}(t) = \frac{S_2 S_3 S_5 S_8 \kappa_n I_{op}^2}{S_4 S_6 CV_T} e^{\frac{\kappa_p (Vdd - V_w)}{V_T}} \tag{10}$$

In terms of the general solution expressed in (3), we have

$$\tau = \frac{CV_T}{\kappa_n I_\tau} \tag{11}$$

$$i_{syn}(0) = S_8 I_{0n} e^{\frac{\kappa_n v_c(0)}{V_T}} \tag{12}$$

$$i_{syn}(\infty) = \frac{S_2 S_3 S_5 S_8 I_{op}^2}{S_4 S_6 I_\tau} e^{\frac{\kappa_p(Vdd-Vw)}{V_T}} \tag{13}$$

When the spike input is off ($t > t_{pw}$) and we neglect the leakage current from $M_2$, then $i_{syn}(t)$ will exponentially decay with the same time constant defined by (11). That is,

$$i_{syn}(t) = i_{syn}(t_{pw}) e^{-\frac{(t-t_{pw})}{\tau}} \tag{14}$$

## 4 Results

### 4.1 Comparison of theory and measurement

We have fabricated a chip containing the basic synapse circuit as shown in Figure 2 through MOSIS in a commercially-available 1.5 $\mu$m, double poly fabrication process. In order to compare our theoretical prediction with chip measurement, we first estimate the two transistor parameters $\kappa$ and $I_0$ by measuring the drain currents from test transistors on the same chip. The current measurements were performed with a Keithley 6517A electrometer. $\kappa$ and $I_0$ are estimated by fitting Equation (5) (and PMOS with PMOS i-v equation) through multiple measurements of (vgs, ids) points through linear regression. The two parameters are found to be $\kappa_n = 0.67$, $I_{0n} = 1.32 \times 10^{-14} A$, $\kappa_p = 0.77$, $I_{0p} = 1.33 \times 10^{-19} A$. In estimating these two parameters as well as to compute our model predictions, we estimate the effective transistor width for the wide transistors (e.g. $M_8$ with m=20).

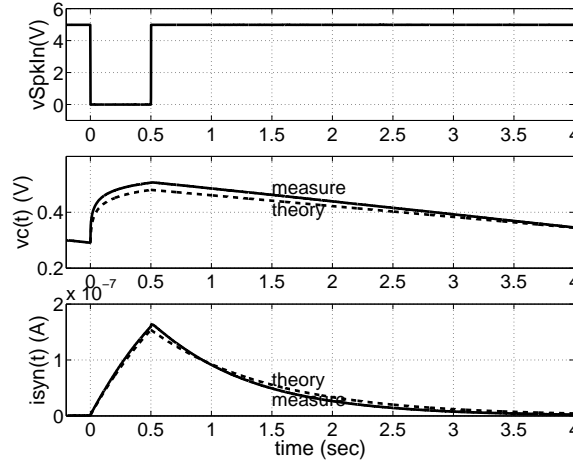

Figure 3: Comparison between model prediction and measurement. To illustrate the detailed time course, we used a large spike pulse width. We set $V_\tau = 0$ and $V_w = 3.85V$.

Figure 3 illustrates our test results compared against the model prediction. We used a very wide pulse to exaggerate the details in the time response. Note that as the time constant is so large, the $i_{syn}(t)$ rises almost linearly during the spike. In this case, $V_w = 3.85V$.

### 4.2 Tuning of synaptic strength and time constant

The synaptic time constant is solely determined by the leak current through transistor $M_7$. The control is achieved by turning the pin $V_\tau$. The synaptic strength is controlled by Vw (which is also coupled with $I_\tau$) as can be seen from (13). In Figure 4, we present our test results that illustrate how the various time constants and synaptic strengths can be achieved.

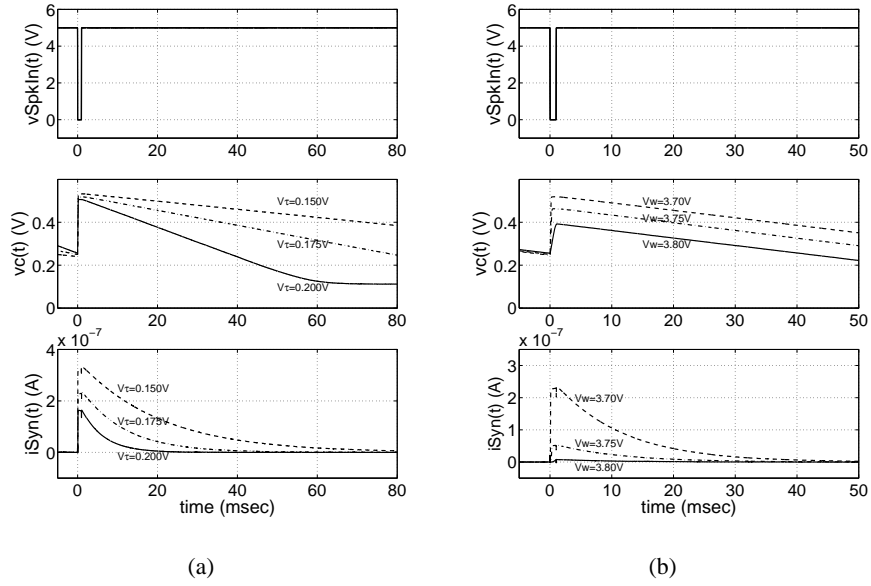

Figure 4: Changing time constant $\tau$ and synaptic strength. (a) Keeping $V_w = 3.7V$ constant, but changing $V_\tau$. (b) Keeping $V_\tau = 0.175V$, but changing $V_w$. In both (a) and (b), spike pulse width is set as 1 msec.

### 4.3 Spike train response

The exponential rise of the synaptic current during a spike naturally provides the summation and saturation of incoming spikes. Figure 5 illustrates this behavior in response to an input spike train of fixed duration.

## 5 Discussion

We have proposed a new synapse model and a specific CMOS implementation of the model. In our theoretical analysis, we have ignored all parasitic effects which can play an significant role in the circuit behavior. For example, as the source follower $M_3 - M_4$ provides the gate voltage of $M_2$, switching through $M_1$ will affect the circuit behavior due to parasitic capacitance. We emphasize that various circuit implementation can be designed, especially a circuit with lower glitch but faster speed is preferred.

The synaptic model circuit we have described has a single time constant for both its rising and decaying phase, whereas the time-course of biological synapses show a faster rising phase, but a much slower decaying phase. The second time constant can, in principle, be implemented in our circuit by adding a parallel branch to $M_7$ with some switching circuitry.

Biological synapses have been best modeled and fitted by an exponentially-decaying time course with different time constants for different types of synapse. Our synapse circuit model captures this important characteristic of the biological synapse, providing an easily controlled exponential decay and a natural summation and saturation of the synaptic current. By using a simple first order linear model, our synapse circuit model can give the circuit designer an analytically tractable function for use in large, complex, spiking neural network system design. The current mirror synapse, in spite of its successful application,

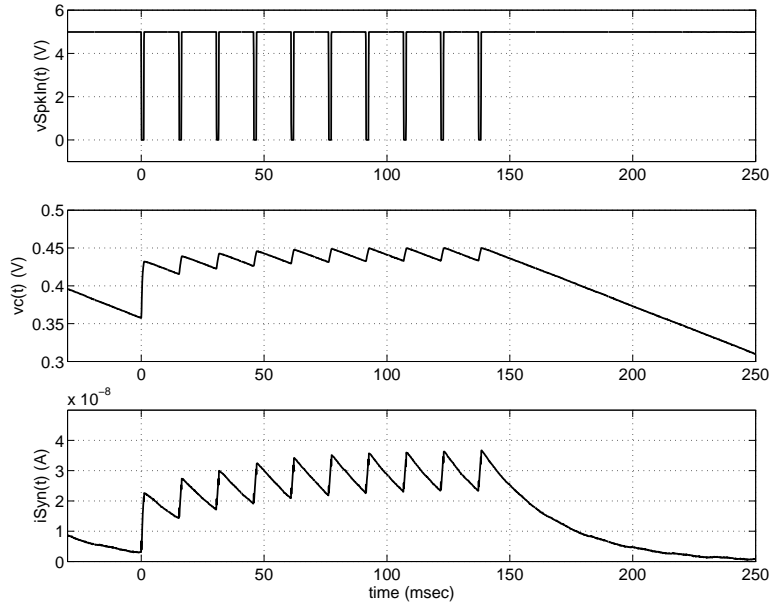

Figure 5: Response to spike train. The spike pulse width is set as 1 msec, and period 15 msec. $V_w = 3.73V$, $V_\tau = 131mV$.

has been found to be an inconvenient computation unit due to its nonlinearity. Our linear synapse is achieved, however, with the cost of silicon size. This is especially true when utilized in an AER system, where the spike can be less than a microsecond. Because our linearity is achieved by employing the CMOS subthreshold current characteristic, working with very narrow pulses will mean the use of large transistor widths to get large charging currents. We have identified a number of modifications that may allow the circuit to operate at much higher current levels and thus higher speed.

## 6  Conclusion

We have identified a need for more independent control of the synaptic gain, time-course, and leak parameters in CMOS synapse and have demonstrated a prototype circuit that utilizes current-mode feedback to exhibit the same first-order dynamics that are utilized by Destexhe et al. [14], [15] to describe a kinetic model description of receptor-neurotransmitter binding for a more efficient computational description of the synaptic conductance. The specific implementation relies on the subthreshold exponential characteristic of the MOSFET and thus operates best at these current levels and slower speeds.

**Acknowledgments**

This work was supported by funding from DARPA (N0001400C0315) and the Air Force Office of Strategic Research (AFOSR - F496200110415). We thank MOSIS for fabrication services in support of our neuromorphic analog VLSI course and teaching laboratory.

## References

[1]  M. Mahowald, *An Analog VLSI System for Stereoscopic Vision.*   Norwell, MA: Kluwer Academic, 1994.

[2] A. Mortara, "A pulsed communication/computation framework for analog VLSI perceptive systems," in *Neuromorphic Systems Engineering*, T. S. Land, Ed. Norwell, MA: Kluwer Academic Publishers, 1998, pp. 217–228.

[3] S. Deiss, R. Douglas, and A. Whatley, "A pulse-coded communications infrastructure for neuromorphic systems," in *Pulsed Neural Networks*, W. Mass and C. Bishop, Eds. Cambridge, MA: MIT Press, 1999, pp. 157–178.

[4] K. A. Boahen, "The retinomorphic approach: adaptive pixel-parallel amplification, filtering, and quantization," *Journal of Analog Integrated Circuits and Signal Processing*, vol. 13, pp. 53–68, 1997.

[5] M. Cheely and T. Horiuchi, "Analog VLSI models of range-tuned neurons in the bat echolocation system," *EURASIP Journal, Special Issue on Neuromorphic Signal Processing and Implementations (in press)*, 2003.

[6] C. Rasche and R. H. R. Hahnloser, "Silicon synaptic depression," *Biol. Cybern.*, vol. 84, pp. 57–62, 2001.

[7] A. McEwan and A. van Schaik, "A silicon representation of the Meddis inner hair cell model," in *Proceedings of the ICSC Symposia on Intelligent Systems & Application (ISA'2000)*, 2000, paper 1544-078.

[8] M. Boegerhausen, P. Suter, and S. Liu, "Modeling short-term synaptic depression in silicon," *Neural Computation*, vol. 15, no. 2, pp. 331–348, Feb 2003.

[9] P. Hafliger, M. Mahowald, and L.Watts, "A spike based learning neuron in analog VLSI," in *Advances in Neural Information Processing Systems*, M. C. Mozer, M. I. Jordan, and T. Petsche, Eds. Cambridge, MA: MIT Press, 1997, vol. 9, pp. 692–698.

[10] G. Indiveri, "Neuromorphic bistable VLSI synapses with spike-timing-dependent plasticity," in *Advances in Neural Information Processing Systems*, M. C. Mozer, M. I. Jordan, and T. Petsche, Eds. Cambridge, MA: MIT Press, 2002, vol. 15.

[11] J. P. Lazzaro, "Low-power silicon axons, neuons, and synapses," in *Silicon Implementations of Pulse Coded Neural Networks*, M. E. Zaghloul, J. L. Meador, and R. W. Newcomb, Eds. Norwell, MA: Kluwer Academic Publishers, 1994, pp. 153–164.

[12] W. Gerstner, *Spiking Neuron Models: Single Neurons, Populations, Plasticity*. Cambridge, UK: Cambridge Unvisity Press, 2002.

[13] W. Rall, "Distinguishing theoretical synaptic potentials computed for different soma-dendritic distributions of synaptic inputs," *J. Neurophys.*, vol. 30, pp. 1138–1168, 1967.

[14] A. Destexhe, Z. F. Mainen, and T. J. Sejnowski, "Synthesis of models for excitable membranes, synaptic transmission and neuromodulation using a common kinetic formalism," *Journal of Computational Neuroscience*, vol. 1, pp. 195–230, 1994.

[15] ——, "An efficient method for computing synaptic conductances based on a kinetic model of receptor binding," *Neural Computation*, vol. 6, pp. 14–18, 1994.

[16] E. Seevinck, "Companding current-mode integrator: A new circuit principle for continuous time monolithic filters," *Electron. Letts.*, vol. 26, pp. 2046–2047, Nov 1990.

[17] D. R. Frey, "Exponential state space fitlers: A generic current mode design strategy," *IEEE Trans. Circuits Syst. I*, vol. 43, pp. 34–42, Jan 1996.

[18] Y. Tsividis, "Externally linear, time-invariant systems and their application to companding signal processors," *IEEE Trans. Circuits Syst. II*, vol. 44, pp. 65–85, Feb 1997.

[19] C. Mead, *Analog VLSI and Neural Systems*. Reading, MA: Addison-Wesley, 1989.

[20] E. A. Vittoz and J. Fellrath, "CMOS analog integrated circuits based on weak inversion opearaton," *IEEE J. Solid-State Circuits*, vol. 12, pp. 224–231, Jun. 1977.

[21] S.-C. Liu, J. Kramer, G. Indiveri, T. Delbruck, and R. Douglas, *Analog VLSI: Circuits and Principle*. Cambridge, MA: The MIT Press, 2002.
